# Experimental Results on Learning Stochastic Memoryless Policies for Partially Observable Markov Decision Processes

**John K. Williams**
Department of Mathematics
University of Colorado
Boulder, CO 80309-0395
jkwillia@euclid.colorado.edu

**Satinder Singh**
AT&T Labs-Research
180 Park Avenue
Florham Park, NJ 07932
baveja@research.att.com

## Abstract

Partially Observable Markov Decision Processes (POMDPs) constitute an important class of reinforcement learning problems which present unique theoretical and computational difficulties. In the absence of the Markov property, popular reinforcement learning algorithms such as $Q$-learning may no longer be effective, and memory-based methods which remove partial observability via state-estimation are notoriously expensive. An alternative approach is to seek a stochastic memoryless policy which for each observation of the environment prescribes a probability distribution over available actions that maximizes the average reward per timestep. A reinforcement learning algorithm which learns a locally optimal stochastic memoryless policy has been proposed by Jaakkola, Singh and Jordan, but not empirically verified. We present a variation of this algorithm, discuss its implementation, and demonstrate its viability using four test problems.

## 1 INTRODUCTION

Reinforcement learning techniques have proven quite effective in solving Markov Decision Processes (MDPs), control problems in which the exact state of the environment is available to the learner and the expected result of an action depends only on the present state [10]. Algorithms such as $Q$-learning learn optimal deterministic policies for MDPs—rules which for every state prescribe an action that maximizes the expected future reward. In many important problems, however, the exact state of the environment is either inherently unknowable or prohibitively expensive to obtain, and only a limited, possibly stochastic observation of the environment is available. Such

Partially Observable Markov Decision Processes (POMDPs) [3,6] are often much more difficult than MDPs to solve [4]. Distinct sequences of observations and actions preceding a given observation in a POMDP may lead to different probabilities of occupying the underlying exact states of the MDP. If the efficacy of an action depends on the hidden exact state of the environment, an optimal choice may require knowing the past history as well as the current observation, and the problem is no longer Markov. In light of this difficulty, one approach to solving POMDPs is to explore the environment while building up a memory of past observations, actions and rewards which allows estimation of the current hidden state [1]. Such methods produce deterministic policies, but they are computationally expensive and may not scale well with problem size. Furthermore, policies that require state-estimation using memory may be complicated to implement.

Memoryless policies are particularly appropriate for problems in which the state is expensive to obtain or inherently difficult to estimate, and they have the advantage of being extremely simple to act upon. For a POMDP, the optimal memoryless policy is generally a stochastic policy—one which for each observation of the environment prescribes a probability distribution over the available actions. In fact, examples of POMDPs can be constructed for which a stochastic policy is arbitrarily better than the optimal deterministic policy [9]. An algorithm proposed by Jaakkola, Singh and Jordan (JSJ) [2], which we investigate here, learns memoryless stochastic policies for POMDPs.

## 2   POMDPs AND DIFFERENTIAL-REWARD $Q$-VALUES

We assume that the environment has discrete states $S = \{s^1, s^2, .. s^N\}$, and the learner chooses actions from a set $\mathcal{A}$. State transitions depend only on the current state $s$ and the action $a$ taken (the Markov property); they occur with probabilities $P^a(s,s')$ and result in expected rewards $R^a(s,s')$. In a POMDP, the learner cannot sense exactly the state $s$ of the environment, but rather perceives only an observation—or "message"—from a set $\mathcal{M} = \{m^1, m^2, .. m^M\}$ according to a conditional probability distribution $P(m|s)$. The learner will in general not know the size of the underlying state space, its transition probabilities, reward function, or the conditional distributions of the messages.

In MDPs, there always exists a policy which simultaneously maximizes the expected future reward for all states, but this is not the case for POMDPs [9]. An appropriate alternative measure of the merit of a stochastic POMDP policy $\pi(a|m)$ is the asymptotic average reward per timestep, $R^\pi$, that it achieves. In seeking an optimal stochastic policy, the JSJ algorithm makes use of $Q$-values determined by the infinite-horizon differential reward for each observation-action pair $(m,a)$. In particular, if $r_t$ denotes the reward obtained at time $t$, we may define the differential-reward $Q$-values by

$$Q^\pi(s,a) = \sum_{t=1}^{\infty} E_\pi[r_t - R^\pi \mid s_1 = s, a_1 = a] ; \; Q^\pi(m,a) = E_s[Q^\pi(s,a) \mid M(s) = m] \quad (1)$$

where $M$ is the observation operator. Note that $E[r_t] \to R^\pi$ as $t \to \infty$, so the summand converges to zero. The value functions $V^\pi(s)$ and $V^\pi(m)$ may be defined similarly.

## 3   POLICY IMPROVEMENT

The JSJ algorithm consists of a method for evaluating $Q^\pi$ and $V^\pi$ and a mechanism for using them to improve the current policy. Roughly speaking, if $Q^\pi(m,a) > V^\pi(m)$, then action $a$ realized a higher differential reward than the average for observation $m$, and assigning it a slightly greater probability will increase the average reward per timestep, $R^\pi$. We interpret the quantities $\Delta_m(a) = Q^\pi(m,a) - V^\pi(m)$ as comprising a "gradient" of $R^\pi$ in policy space. Their projections onto the probability simplexes may then be written

as $\delta_m = \Delta_m - <\Delta_m, \mathbf{1}> \mathbf{1}/|\mathcal{A}|$, where $\mathbf{1}$ is the one-vector $(1,1,...,1)$, $<,>$ is the inner product, and $|\mathcal{A}|$ is the number of actions, or

$$\delta_m(a) = \Delta_m(a) - \frac{1}{|A|}\sum_{a' \in A}\Delta_m(a') = Q^\pi(m,a) - \frac{1}{|A|}\sum_{a' \in A}Q^\pi(m,a'). \tag{2}$$

For sufficiently small $\varepsilon_m$, an improved policy $\pi'(a|m)$ may be obtained by the increments

$$\pi'(a|m) = \pi(a|m) + \varepsilon_m\,\delta_m(a). \tag{3}$$

In practice, we also enforce $\pi'(a|m) \geq P_{min}$ for all $a$ and $m$ to guarantee continued exploration. The original JSJ algorithm prescribed using $\Delta_m(a)$ in place of $\delta_m(a)$ in equation (3), followed by renormalization [2]. Our method has the advantage that a given value of $\Delta$ yields the same increment regardless of the current value of the policy, and it ensures that the step is in the correct direction. We also do not require the differential-reward value estimate, $V^\pi$.

# 4  $Q$-EVALUATION

As the POMDP is simulated under a fixed stochastic policy $\pi$, every occurrence of an observation-action pair $(m,a)$ begins a sequence of rewards which can be used to estimate $Q^\pi(m,a)$. Exploiting the fact that the $Q^\pi(m,a)$ are defined as sums, the JSJ $Q$-evaluation method recursively averages the estimates from all such sequences using a so-called "every-visit" Monte-Carlo method. In order to reduce the bias and variance caused by the dependence of the evaluation sequences, a factor $\beta$ is used to discount their shared "tails". Specifically, at time $t$ the learner makes observation $m_t$, takes action $a_t$, and obtains reward $r_t$. The number of visits $K(m_t, a_t)$ is incremented, the tail discount rate $\gamma(m,a) = 1 - K(m,a)^{-1/4}$, and the following updates are performed (the indicator function $\chi_t(m,a)$ is 1 if $(m,a) = (m_t, a_t)$ and 0 otherwise).

$$\beta(m,a) = \left[1 - \frac{\chi_t(m,a)}{K(m,a)}\right]\gamma(m,a)\beta(m,a) + \frac{\chi_t(m,a)}{K(m,a)} \quad \text{(tail discount factor)} \tag{4}$$

$$Q(m,a) = \left[1 - \frac{\chi_t(m,a)}{K(m,a)}\right]Q(m,a) + \beta(m,a)\,[r_t - R] \quad \text{($Q^\pi$-estimate)} \tag{5}$$

$$C(m,a) = \left[1 - \frac{\chi_t(m,a)}{K(m,a)}\right]C(m,a) + \beta(m,a) \quad \text{(cumulative discount effect)} \tag{6}$$

$$R = (1 - 1/t)R + (1/t)\,r_t \quad \text{($R^\pi$-estimate)} \tag{7}$$

$$Q(m,a) = Q(m,a) - C(m,a)\,[R - R_{old}];\ R_{old} = R \quad \text{($Q^\pi$-estimate correction)} \tag{8}$$

Other schedules for $\gamma(m,a)$ are possible—see [2]—and the correction provided by (8) need not be performed at every step, but can be delayed until the $Q^\pi$-estimate is needed.

This evaluation method can be used as given for a policy-iteration type algorithm in which independent $T$-step evaluations of $Q^\pi$ are interspersed with policy improvements as prescribed in section 3. However, an online version of the algorithm which performs policy improvement after every step requires that old experience be gradually "forgotten" so that the $Q^\pi$-estimate can respond to more recent experience. To achieve this, we multiply the previous estimates of $\beta$, $Q$, and $C$ at each timestep by a "decay" factor $\alpha$, $0 < \alpha < 1$, before they are updated via equations (4)–(6), and replace equation (7) by

$$R = \alpha(1 - 1/t)R + [1 - \alpha(1 - 1/t)]\,r_t. \tag{9}$$

An alternative method, which also works reasonably well, is to multiply $K$ and $t$ by $\alpha$ at each timestep instead.

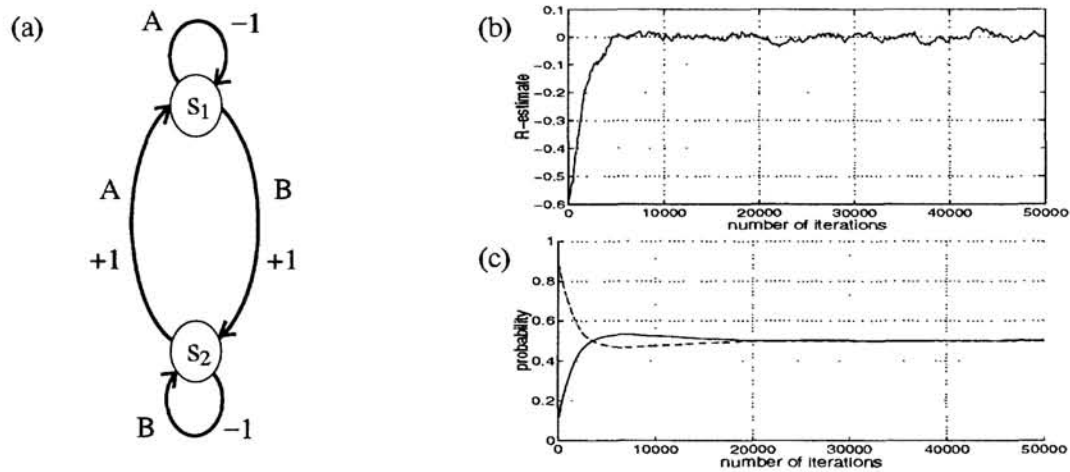

Figure 1: (a) Schematic of confounded two-state POMDP, (b) evolution of the $R^{\pi}$-estimate, and (c) evolution of $\pi(A)$ *(solid)* and $\pi(B)$ *(dashed)* for $\varepsilon = 0.0002$, $\alpha = 0.9995$.

## 5   EMPIRICAL RESULTS

We present only results from single runs of our online algorithm, including the modified JSJ policy improvement and $Q$-evaluation procedures described above. Results from the policy iteration version are qualitatively similar, and statistics performed on multiple runs verify that those shown are representative of the algorithm's behavior. To simplify the presentation, we fix a constant learning rate, $\varepsilon$, and decay factor, $\alpha$, for each problem, and we use $P_{min} = 0.02$ throughout. Note, however, that appropriate schedules or online heuristics for decreasing $\varepsilon$ and $P_{min}$ while increasing $\alpha$ would improve performance and are necessary to ensure convergence. Except for the first problem, we choose the initial policy $\pi$ to be uniform. In the last two problems, values of $\pi(a|m) < 0.03$ are rounded down to zero, with renormalization, before the learned policy is evaluated.

### 5.1   CONFOUNDED TWO-STATE PROBLEM

The two-state MDP diagrammed in Figure 1(a) becomes a POMDP when the two states are confounded into a single observation. The learner may take action A or B, and receives a reward of either +1 or −1; the state transition is deterministic, as indicated in the diagram. Note that either stationary deterministic policy results in $R^{\pi} = -1$, whereas the optimal stochastic policy assigns each action the probability 1/2, resulting in $R^{\pi} = 0$.

The evolution of the $R^{\pi}$-estimate and policy, starting from the initial policy $\pi(A) = 0.1$ and $\pi(B) = 0.9$, is shown in Figure 1. Clearly the learned policy approaches the optimal stochastic policy $\pi = (1/2, 1/2)$.

### 5.2   MATRIX GAME: SCISSORS-PAPER-STONE-GLASS-WATER

Scissors-Paper-Stone-Glass-Water (SPSGW), an extension of the well-known Scissors-Paper-Stone, is a symmetric zero-sum matrix game in which the learner selects a row $i$, the opponent selects a column $j$, and the learner's payoff is determined by the matrix entry $M(i,j)$. A game-theoretic solution is a stochastic (or "mixed") policy which guarantees the learner an expected payoff of at least zero. It can be shown using linear programming that the unique optimal strategy for SPSGW, yielding $R^{\pi} = 0$, is to play stone and water with probability 1/3, and to play scissors, paper, and glass with probability 1/9 [7]. Any stationary deterministic policy results in $R^{\pi} = -1$, since the opponent eventually learns to anticipate the learner's choice and exploit it.

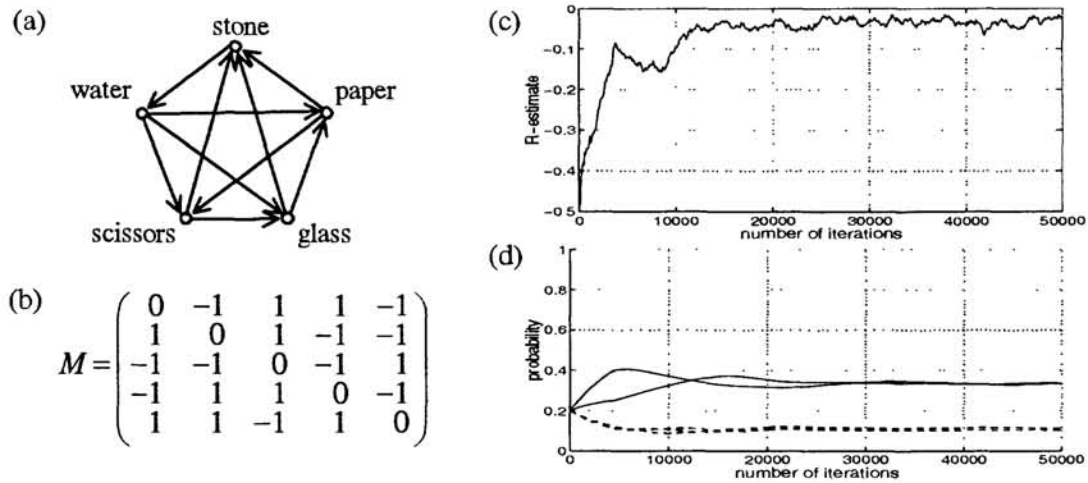

Figure 2: (a) Diagram of Scissors-Paper-Stone-Glass-Water, (b) the payoff matrix, (c) evolution of the $R^\pi$-estimate, and (d) evolution of $\pi$(stone) and $\pi$(water) *(solid)* and $\pi$(scissors), $\pi$(paper), and $\pi$(glass) *(dashed)* for $\varepsilon = 0.00005$, $\alpha = 0.9995$.

In formulating SPSGW as a POMDP, it is necessary to include in the state sufficient information to allow the opponent to exploit any sub-optimal strategy. We thus choose as states the learner's past action frequencies, multiplied at each timestep by the decay factor, $\alpha$. There is only one observation, and the learner acts by selecting the "row" scissors, paper, stone, glass or water, producing a deterministic state transition. The simulated opponent plays the column which maximizes its expected payoff against the estimate of the learner's strategy obtained from the state. The learner's reward is then obtained from the appropriate entry of the payoff matrix.

The policy $\pi = (0.1124, 0.1033, 0.3350, 0.1117, 0.3376)$ learned after 50,000 iterations (see Figure 2) is very close to the optimal policy $\pi = (1/9, 1/9, 1/3, 1/9, 1/3)$.

## 5.3 PARR AND RUSSELL'S GRID WORLD

Parr and Russell's grid world [8] consists of 11 states in a 4x3 grid with a single obstacle as shown in Figure 3(a). The learner senses only walls to its immediate east or west and whether it is in the goal state (upper right corner) or penalty state (directly below the goal), resulting in the 6 possible observations (0–3, G and P) indicated in the diagram. The available actions are to move N, E, S, or W, but there is a probability 0.1 of slipping to either side and only 0.8 of moving in the desired direction; a movement into a wall results in bouncing back to the original state. The learner receives a reward of +1 for a transition into the goal state, −1 for a transition into the penalty state, and −0.04 for all other transitions. The goal and penalty states are connected to a cost-free absorbing state; when the learner reaches either of them it is teleported immediately to a new start state chosen with uniform probability.

The results are shown in Figure 3. A separate $10^6$-step evaluation of the final learned policy resulted in $R^\pi = 0.047$. In contrast, the optimal deterministic policy indicated by arrows in Figure 3(a) yields $R^\pi = 0.024$ [5], while Parr and Russell's memory-based SPOVA-RL algorithm achieved $R^\pi = 0.12$ after learning for 400,000 iterations [8].

## 5.4 MULTI-SERVER QUEUE

At each timestep, an arriving job having type 1, 2, or 3 with probability 1/2, 1/3 or 1/6, respectively, must be assigned to server A, B or C; see Figure 4(a). Each server is optimized for a particular job type which it can complete in an expected time of 2.6

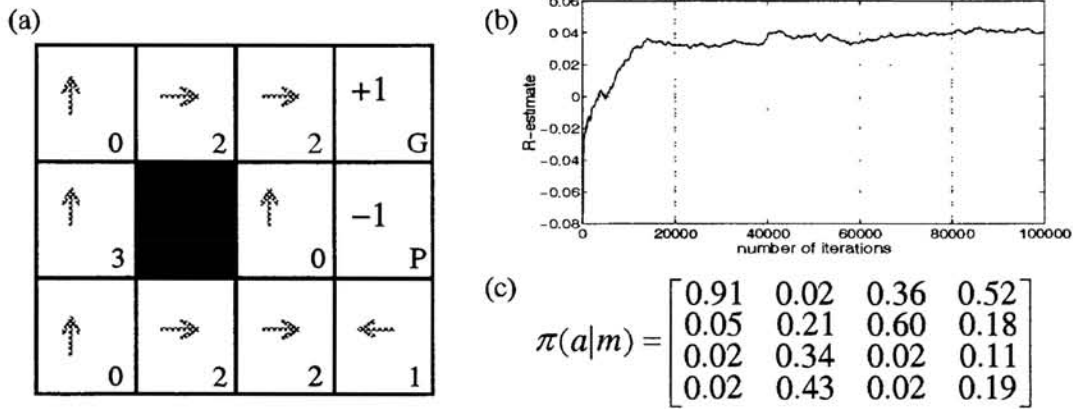

Figure 3: (a) Parr and Russell's grid world, with observations shown in lower right corners and the optimal deterministic memoryless policy represented by arrows, (b) evolution of the $R^\pi$-estimate, and (c) the resulting learned policy (observations 0–3 across columns, actions N, E, S, W down rows) for $\varepsilon = 0.02$, $\alpha = 0.9999$.

timesteps, while the other job types require 50% longer. All jobs in a server's queue are handled in parallel, up to a capacity of 10 for each server; they finish with probability $1/f$ at each timestep, where $f$ is the product of the expected time for the job and the number of jobs in the server's queue. The states for this POMDP are all combinations of waiting jobs and server occupancies of the three job types, but the learner's observation is restricted to the type of the waiting job. The state transition is obtained by removing all jobs which have finished and adding the waiting job to the chosen server if it has space available. The reward is +1 if the job is successfully placed, or 0 if it is dropped.

The results are shown in Figure 4. A separate $10^6$-step evaluation of the learned policy obtained $R^\pi = 0.95$, corresponding to 95% success in placing jobs. In contrast, the optimal deterministic policy, which assigns each job to the server optimized for it, attained only 87% success. Thus the learned policy more than halves the drop rate!

## 6   CONCLUSION

Our online version of an algorithm proposed by Jaakkola, Singh and Jordan efficiently learns a stochastic memoryless policy which is either provably optimal or at least superior to any deterministic memoryless policy for each of four test problems. Many enhancements are possible, including appropriate learning schedules to improve performance and ensure convergence, estimation of the time between observation-action visits to obtain better discount rates $\gamma$ and thereby enhance $Q^\pi$-estimate bias and variance reduction (see [2]), and multiple starts or simulated annealing to avoid local minima. In addition, observations could be extended to include some past history when appropriate.

Most POMDP algorithms use memory and attempt to learn an optimal deterministic policy based on belief states. The stochastic memoryless policies learned by the JSJ algorithm may not always be as good, but they are simpler to act upon and can adapt smoothly in non-stationary environments. Moreover, because it searches the space of stochastic policies, the JSJ algorithm has the potential to find the optimal memoryless policy. These considerations, along with the success of our simple implementation, suggest that this algorithm may be a viable candidate for solving real-world POMDPs, including distributed control or network admission and routing problems in which the numbers of states are enormous and complete state information may be difficult to obtain or estimate in a timely manner.

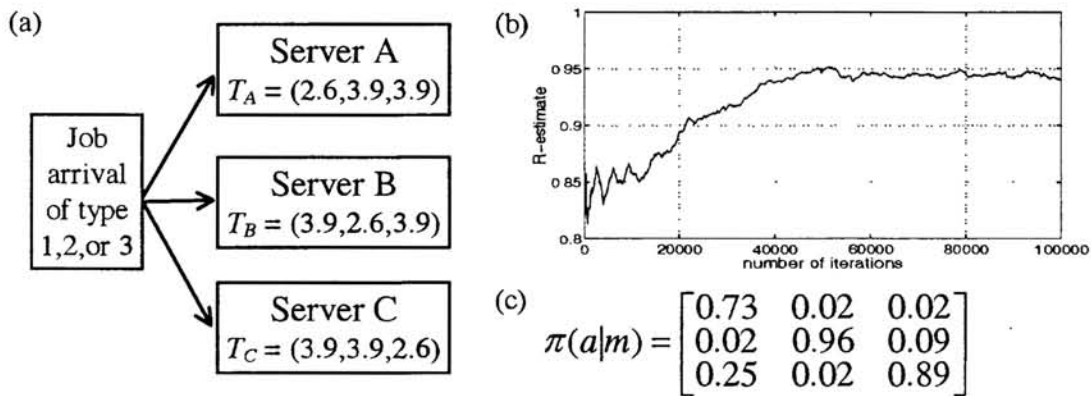

Figure 4: (a) Schematic of the multi-server queue, (b) evolution of the $R^\pi$-estimate, and (c) the resulting learned policy (observations 1, 2, 3 across columns, actions A, B, C down rows) for $\varepsilon = 0.005$, $\alpha = 0.9999$.

## Acknowledgements

We would like to thank Mike Mozer and Tim Brown for helpful discussions. Satinder Singh was funded by NSF grant IIS-9711753.

## References

[1] Chrisman, L. (1992). Reinforcement learning with perceptual aliasing: The perceptual distinctions approach. In *Proceedings of the Tenth National Conference on Artificial Intelligence*.

[2] Jaakkola, T., Singh, S. P., and Jordan, M. I. (1995). Reinforcement learning algorithm for partially observable Markov decision problems. In *Advances in Neural Information Processing Systems 7*.

[3] Littman, M., Cassandra, A., and Kaelbling, L. (1995). Learning policies for partially observable environments: Scaling up. In *Proceedings of the Twelfth International Conference on Machine Learning*.

[4] Littman, M. L. (1994). Memoryless policies: Theoretical limitations and practical results. *Proceedings of the Third International Conference on Simulation of Adaptive Behavior: From Animals to Animats*.

[5] Loch, J., and Singh, S. P. (1998). Using eligibility traces to find the best memoryless policy in partially observable Markov decision processes. In *Machine Learning: Proceedings of the Fifteenth International Conference*.

[6] Lovejoy, W. S. (1991). A survey of algorithmic methods for partially observable Markov decision processes. In *Annals of Operations Research, 28*.

[7] Morris, P. (1994). *Introduction to Game Theory*. Springer-Verlag, New York.

[8] Parr, R. and Russell, S. (1995). Approximating optimal policies for partially observable stochastic domains. In *Proceedings of the International Joint Conference on Artificial Intelligence*.

[9] Singh, S. P., Jaakkola, T., and Jordan, M. I. (1994). Learning without state-estimation in partially observable Markovian decision processes. In *Machine Learning: Proceedings of the Eleventh International Conference*.

[10] Sutton, R. S. and Barto, A. G. (1998). *Reinforcement Learning: An Introduction*. MIT Press.

